# Fast Gaussian Process Regression using KD-Trees

**Yirong Shen**
Electrical Engineering Dept.
Stanford University
Stanford, CA 94305

**Andrew Y. Ng**
Computer Science Dept.
Stanford University
Stanford, CA 94305

**Matthias Seeger**
Computer Science Div.
UC Berkeley
Berkeley, CA 94720

## Abstract

The computation required for Gaussian process regression with $n$ training examples is about $O(n^3)$ during training and $O(n)$ for each prediction. This makes Gaussian process regression too slow for large datasets. In this paper, we present a fast approximation method, based on kd-trees, that significantly reduces both the prediction and the training times of Gaussian process regression.

## 1 Introduction

We consider (regression) estimation of a function $\boldsymbol{x} \mapsto u(\boldsymbol{x})$ from noisy observations. If the data-generating process is not well understood, simple parametric learning algorithms, for example ones from the generalized linear model (GLM) family, may be hard to apply because of the difficulty of choosing good features. In contrast, the nonparametric *Gaussian process (GP)* model [19] offers a flexible and powerful alternative. However, a major drawback of GP models is that the computational cost of learning is about $O(n^3)$, and the cost of making a single prediction is $O(n)$, where $n$ is the number of training examples. This high computational complexity severely limits its scalability to large problems, and we believe has proved a significant barrier to the wider adoption of the GP model.

In this paper, we address the scaling issue by recognizing that learning and predictions with a GP regression (GPR) model can be implemented using the matrix-vector multiplication (MVM) primitive $\boldsymbol{z} \mapsto \boldsymbol{K}\boldsymbol{z}$. Here, $\boldsymbol{K} \in \mathbb{R}^{n,n}$ is the *kernel matrix*, and $\boldsymbol{z} \in \mathbb{R}^n$ is an arbitrary vector. For the wide class of so-called *isotropic* kernels, MVM can be approximated efficiently by arranging the dataset in a tree-type multiresolution data structure such as *kd-trees* [13], *ball trees* [11], or *cover trees* [1]. This approximation can sometimes be made orders of magnitude faster than the direct computation, without sacrificing much in terms of accuracy.

Further, the storage requirements for the tree is $O(n)$, while a direct storage of the kernel matrix would require $O(n^2)$ spare. We demonstrate the efficiency of the tree approach on several large datasets.

In the sequel, for the sake of simplicity we will focus on kd-trees (even though it is known that kd-trees do not scale well to high dimensional data). However, it is also completely straightforward to apply the ideas in this paper to other tree-type data structures, for example ball trees and cover trees, which typically scale significantly better to high dimensional data.

## 2 The Gaussian Process Regression Model

Suppose that we observe some data $D = \{(\boldsymbol{x}_i, y_i) \,|\, i = 1, \ldots, n\}$, $\boldsymbol{x}_i \in \mathcal{X}$, $y_i \in \mathbb{R}$, sampled independently and identically distributed (i.i.d.) from some unknown distribution.

Our goal is to predict the response $y_*$ on future test points $\boldsymbol{x}_*$ with small mean-squared error under the data distribution. Our model consists of a latent (unobserved) function $\boldsymbol{x} \mapsto u$ so that $y_i = u_i + \varepsilon_i$, where $u_i = u(\boldsymbol{x}_i)$, and the $\varepsilon_i$ are independent Gaussian noise variables with zero mean and variance $\sigma^2 > 0$. Following the Bayesian paradigm, we place a prior distribution $P(u(\cdot))$ on the function $u(\cdot)$ and use the posterior distribution

$$P(u(\cdot)|D) \propto N(\boldsymbol{y}|\boldsymbol{u}, \sigma^2 \boldsymbol{I})P(u(\cdot))$$

in order to predict $y_*$ on new points $\boldsymbol{x}_*$. Here, $\boldsymbol{y} = [y_1, \ldots, y_n]^T$ and $\boldsymbol{u} = [u_1, \ldots, u_n]^T$ are vectors in $\mathbb{R}^n$, and $N(\cdot|\mu, \Sigma)$ is the density of a Gaussian with mean $\mu$ and covariance $\Sigma$. For a GPR model, the prior distribution is a (zero-mean) Gaussian process defined in terms of a positive definite kernel (or covariance) function $K : \mathcal{X}^2 \to \mathbb{R}$. For the purposes of this paper, a GP can be thought of as a mapping from arbitrary finite subsets $\{\tilde{\boldsymbol{x}}_i\} \subset \mathcal{X}$ of points, to corresponding zero-mean Gaussian distributions with covariance matrix $\tilde{\boldsymbol{K}} = (K(\tilde{\boldsymbol{x}}_i, \tilde{\boldsymbol{x}}_j))_{i,j}$. (This notation indicates that $\tilde{\boldsymbol{K}}$ is a matrix whose $(i, j)$-element is $K(\tilde{\boldsymbol{x}}_i, \tilde{\boldsymbol{x}}_j)$.) In this paper, we focus on the problem of speeding up GPR under the assumption that the kernel is monotonic isotropic. A kernel function $K(\boldsymbol{x}, \boldsymbol{x}')$ is called *isotropic* if it depends only on the Euclidean distance $r = \|\boldsymbol{x} - \boldsymbol{x}'\|_2$ between the points, and it is *monotonic isotropic* if it can be written as a monotonic function of $r$.

## 3  Fast GPR predictions

Since $u(\boldsymbol{x}_1), u(\boldsymbol{x}_2), \ldots, u(\boldsymbol{x}_n)$ and $u(\boldsymbol{x}_*)$ are jointly Gaussian, it is easy to see that the predictive (posterior) distribution $P(u_*|D)$, $u_* = u(\boldsymbol{x}_*)$ is given by

$$P(u_*|D) = N\left(u_* \mid \boldsymbol{k}_*^T \boldsymbol{M}^{-1}\boldsymbol{y}, \; K(\boldsymbol{x}_*, \boldsymbol{x}_*) - \boldsymbol{k}_*^T \boldsymbol{M}^{-1}\boldsymbol{k}_*\right), \tag{1}$$

where $\boldsymbol{k}_* = [K(\boldsymbol{x}_*, \boldsymbol{x}_1), \ldots, K(\boldsymbol{x}_*, \boldsymbol{x}_n)]^T \in \mathbb{R}^n$, and $\boldsymbol{M} = \boldsymbol{K} + \sigma^2 \boldsymbol{I}$, $\boldsymbol{K} = (K(\boldsymbol{x}_i, \boldsymbol{x}_j))_{i,j}$. Therefore, if $\boldsymbol{p} = \boldsymbol{M}^{-1}\boldsymbol{y}$, the optimal prediction under the model is $\hat{u}_* = \boldsymbol{k}_*^T \boldsymbol{p}$, and the predictive variance (of $P(u_*|D)$) can be used to quantify our uncertainty in the prediction. Details can be found in [19]. ([16] also provides a tutorial on GPs.)

Once $\boldsymbol{p}$ is determined, making a prediction now requires that we compute

$$\boldsymbol{k}_*^T \boldsymbol{p} = \sum_{i=1}^n K(\boldsymbol{x}_*, \boldsymbol{x}_i)p_i = \sum_{i=1}^n w_i p_i \tag{2}$$

which is $O(n)$ since it requires scanning through the entire training set and computing $K(\boldsymbol{x}_*, \boldsymbol{x}_i)$ for each $\boldsymbol{x}_i$ in the training set. When the training set is very large, this becomes prohibitively slow. In such situations, it is desirable to use a fast approximation instead of the exact direct implementation.

### 3.1  Weighted Sum Approximation

The computations in Equation 2 can be thought of as a weighted sum, where $w_i = K(\boldsymbol{x}_*, \boldsymbol{x}_i)$ is the weight on the i-th summand $p_i$. We observe that if the dataset is divided into groups where all data points in a group have similar weights, then it is possible to compute a fast approximation to the above weighted sum. For example, let $G$ be a set of data points that all have weights near some value $w$. The contribution to the weighted sum by points in $G$ is

$$\sum_{i:\boldsymbol{x}_i \in G} w_i p_i = \sum_{i:\boldsymbol{x}_i \in G} w p_i + \sum_{i:\boldsymbol{x}_i \in G} (w_i - w)p_i = w \sum_{i:\boldsymbol{x}_i \in G} p_i + \sum_{i:\boldsymbol{x}_i \in G} \epsilon_i p_i$$

Where $\epsilon_i = w_i - w$. Assuming that $\sum_{i:\boldsymbol{x}_i \in G} p_i$ is known in advance, $w \sum_{i:\boldsymbol{x}_i \in G} p_i$ can then be computed in constant time and used as an approximation to $\sum_{i:\boldsymbol{x}_i \in G} w_i p_i$ if $\sum_{i:\boldsymbol{x}_i \in G} \epsilon_i p_i$ is small.

We note that for a continuous isotropic kernel function, the weights $w_i = K(\boldsymbol{x}_*, \boldsymbol{x}_i)$ and $w_j = K(\boldsymbol{x}_*, \boldsymbol{x}_j)$ will be similar if $\boldsymbol{x}_i$ and $\boldsymbol{x}_j$ are close to each other. In addition, if the

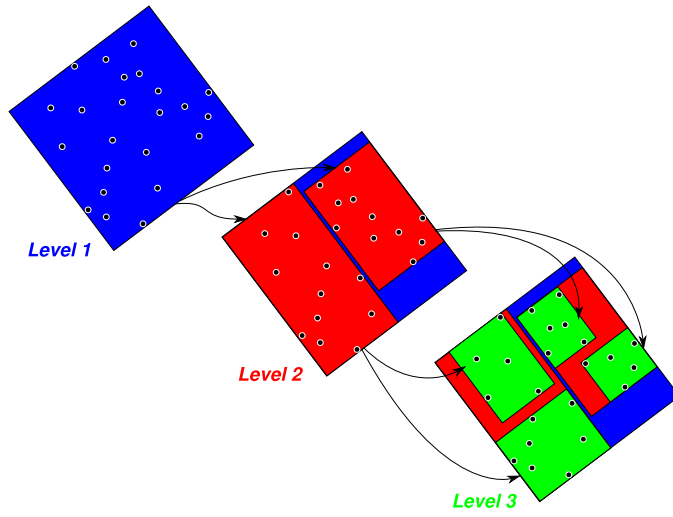

Level 1

Level 2

Level 3

Figure 1: Example of bounding rectangles for nodes in the first three levels of a kd-tree.

kernel function monotonically decreases to zero with increasing $\|\boldsymbol{x}_i - \boldsymbol{x}_j\|$, then points that are far away from the query point $\boldsymbol{x}_*$ will all have weights near zero.

Given a new query, we would like to automatically group points together that have similar weights. But the weights are dependent on the query point and hence the best grouping of the data will also be dependent on the query point. Thus, the problem we now face is, given query point, how to quickly divide the dataset into groups such that data points in the same group have similar weights. Our solution to this problem takes inspiration and ideas from [9], and uses an enhanced kd-tree data structure.

### 3.2   The kd-tree algorithm

A kd-tree [13] is a binary tree that recursively partitions a set of data points. Each node in the kd-tree contains a subset of the data, and records the bounding hyper-rectangle for this subset. The root node contains the entire dataset. Any node that contains more than 1 data point has two child nodes, and the data points contained by the parent node are split among the children by cutting the parent node's bounding hyper-rectangle in the middle of its widest dimension.[1] An example with inputs of dimension 2 is illustrated in Figure 1.

For our algorithm, we will enhance the kd-tree with additional cached information at each node. At a node $\mathbf{ND}$ whose set of data points is $\mathcal{X}_{\mathbf{ND}}$, in addition to the bounding box we also store

1. $N_{\mathbf{ND}} = |\mathcal{X}_{\mathbf{ND}}|$: the number of data points contained by $\mathbf{ND}$.

2. $S_{\mathbf{ND}}^{\text{Unweighted}} = \sum_{\boldsymbol{x}_i \in \mathcal{X}_{\mathbf{ND}}} p_i$: the unweighted sum corresponding to the data contained by $\mathbf{ND}$.

Now, let

$$S_{\mathbf{ND}}^{\text{Weighted}} = \sum_{i:\boldsymbol{x}_i \in \mathcal{X}_{\mathbf{ND}}} K(\boldsymbol{x}_*, \boldsymbol{x}_i) p_i \qquad (3)$$

be the weighted sum corresponding to node $\mathbf{ND}$. One way to calculate $S_{\mathbf{ND}}^{\text{Weighted}}$ is to simply have the 2 children of $\mathbf{ND}$ recursively compute $S_{\text{Left}(\mathbf{ND})}^{\text{Weighted}}$ and $S_{\text{Right}(\mathbf{ND})}^{\text{Weighted}}$ (where

Left(**ND**) and Right(**ND**) are the 2 children of **ND**) and then sum the two results. This takes $O(n)$ time—same as the direct computation—since all $O(n)$ nodes need to be processed. However, if we only want an approximate result for the weighted sum, then we can cut off the recursion at nodes whose data points have nearly identical weights for the given query point.

Since each node maintains a bounding box of the data points that it owns, we can easily bound the maximum weight variation of the data points owned by a node (as in [9]). The nearest and farthest points in the bounding box to the query point can be computed in $O(\text{input dimension})$ operations, and since the kernel function is isotropic monotonic, these points give us the maximum and minimum possible weights $w_{\max}$ and $w_{\min}$ of any data point in the bounding box.

Now, whenever the difference between $w_{\max}$ and $w_{\min}$ is small, we can cutoff the recursion and approximate the weighted sum in Equation 3 by $w * S_{\mathbf{ND}}^{\text{Unweighted}}$ where $w = \frac{1}{2}(w_{\min} + w_{\max})$. The speed and accuracy of the approximation is highly dependent on the cutoff criteria. Moore et al. used the following cutoff rule in [9]:
$$w_{\max} - w_{\min} \le 2\epsilon(W_{\text{SoFar}} + N_{\mathbf{ND}} w_{\min}).$$
Here, $W_{\text{SoFar}}$ is the weight accumulated so far in the computation and $W_{\text{SoFar}} + N_{\mathbf{ND}} w_{\min}$ serves as a lower bound on the total sum of weights involved in the regression. In our experiments, we found that although the above cutoff rule ensures the error incurred at any particular data point in **ND** is small, the total error incurred by all the data points in **ND** can still be high if $N_{\mathbf{ND}}$ is very large. In our experiments (not reported here), their method gave poor performance on the GPR task, in many cases incurring significant errors in the predictions (or, alternatively running no faster than exact computation, if sufficiently small $\epsilon$ is chosen to prevent the large accumulation of errors). Hence, we chose instead the following cutoff rule:
$$N_{\mathbf{ND}}(w_{\max} - w_{\min}) \le 2\epsilon(W_{\text{SoFar}} + N_{\mathbf{ND}} w_{\min}),$$
which also takes into account the total number of points contained in a node.

From the forumla above, we see that the decision of whether to cutoff computation at a node depends on the value of $W_{\text{SoFar}}$ (the total weight of all the points that have been added to the summation so far). Thus it is desirable to quickly accumulate weights at the beginning of the computations, so that more of the later recursions can be cut off. This can be accomplished by going into the child node that's nearer to the query point first when we recurse into the children of a node that doesn't meet the cutoff criteria. (In contrast, [9] always visits the children in left-right order, which in our experiments also gave significantly worse performance than our version.) Our overall algorithm is summarized below:

**WeightedSum**$(\boldsymbol{x}_*, \mathbf{ND}, W_{\text{SoFar}}, \epsilon)$

compute $w_{\max}$ and $w_{\min}$ for the given query point $\boldsymbol{x}_*$
$S_{\mathbf{ND}}^{\text{Weighted}} = 0$
**if** $(w_{\max} - w_{\min}) \le 2\epsilon(W_{\text{SoFar}} + N_{\mathbf{ND}} w_{\min})$
**then**
    $S_{\mathbf{ND}}^{\text{Weighted}} = \frac{1}{2}(w_{\min} + w_{\max}) S_{\mathbf{ND}}^{\text{Unweighted}}$
    $W_{\text{SoFar}} = W_{\text{SoFar}} + w_{\min} N_{\mathbf{ND}}$
    **return** $S_{\mathbf{ND}}^{\text{Weighted}}$
**else**
    determine which child is nearer to the query point $\boldsymbol{x}_*$
    $S_{\text{Nearer}}^{\text{Weighted}} = \textbf{WeightedSum}(\boldsymbol{x}_*, \text{Nearer child of ND}, W_{\text{SoFar}}, \epsilon)$
    $S_{\text{Farther}}^{\text{Weighted}} = \textbf{WeightedSum}(\boldsymbol{x}_*, \text{Farther child of ND}, W_{\text{SoFar}}, \epsilon)$
    $S_{\mathbf{ND}}^{\text{Weighted}} = S_{\text{Nearer}}^{\text{Weighted}} + S_{\text{Farther}}^{\text{Weighted}}$
    **return** $S_{\mathbf{ND}}^{\text{Weighted}}$

## 4 Fast Training

Training (or first-level inference) in the GPR model requires solving the positive definite linear system

$$Mp = y, \quad M = K + \sigma^2 I \qquad (4)$$

for the vector $p$, which in the previous section we assumed had already been pre-computed. Directly calculating $p$ by inverting the matrix $M$ costs about $O(n^3)$ in general. However, in practice there are many ways to quickly obtain approximate solutions to linear systems. Since the system matrix is symmetric positive definite, the *conjugate gradient (CG)* algorithm can be applied. CG is an iterative method which searches for $p$ by maximizing the quadratic function

$$q(z) = y^T z - \frac{1}{2} z^T M z.$$

Briefly, CG ensures that $z$ after iteration $k$ is a maximizer of $q$ over a (Krylow) subspace of dimension $k$. For details about CG and many other approximate linear solvers, see [15]. Thus, $z$ "converges" to $p$ (the unconstrained maximizer of $q$) after $n$ steps, but intermediate $z$ can be used as approximate solutions. The speed of convergence depends on the eigenstructure of $M$. In our case, $M$ typically has only a few large eigenvalues, and most of the spectrum is close to the lower bound $\sigma^2$; under these conditions CG is known to produce good approximations after only a few iterations. Crucially, the only operation on $M$ performed in each iteration of CG is a matrix-vector multiplication (MVM) with $M$.

Since $M = K + \sigma^2 I$, speeding up MVM with $M$ is critically dependent on our ability to perform fast MVM with the kernel matrix $K$. We can apply the algorithm from Section 3 to perform fast MVM.

Specifically, observe that the i-th row of $K$ is given by $k_i = [K(x_i, x_1), \ldots, K(x_i, x_n)]^T$. Thus, $k_i$ has the same form as that of the vector $k_*$ used in the prediction step. Hence to compute the matrix-vector product $Kv$, we simply need to compute the inner products

$$k_i^T v = \sum_{j=1}^{n} K(x_i, x_j) v_j$$

for $i = 1, \ldots, n$. Following exactly the method presented in Section 3, we can do this efficiently using a kd-tree, where here $v$ now plays the role of $p$ in Equation 2.

Two additional optimizations are possible. First, in different iterations of conjugate gradient, we can use the same kd-tree structure to compute $k_i^T v$ for different $i$ and different $v$. Indeed, given a dataset, we need only ever find a single kd-tree structure for it, and the same kd-tree structure can then be used to make multiple predictions or multiple MVM operations. Further, given fixed $v$, to compute $k_i^T v$ for different $i = 1, \ldots, n$ (to obtain the vector resulting from one MVM operation), we can also share the same pre-computed partial unweighted sums in the internal nodes of the tree. Only when $v$ (or $p$) changes do we need to change the partial unweighted sums (discussed in Section 3.2) of $v$ stored in the internal nodes (an $O(n)$ operation).

## 5 Performance Evaluation

We evaluate our kd-tree implementation of GPR and an implementation that uses direct computation for the inner products. Our experiments were performed on the nine regression datasets in Table 1. [2]

| Data set name | Input dimension | Training set size | Test set size |
|---|---|---|---|
| Helicopter yaw rate | 3 | 40000 | 4000 |
| Helicopter $x$-velocity | 2 | 40000 | 4000 |
| Helicopter $y$-velocity | 2 | 40000 | 4000 |
| Mote 10 temperature | 2 | 20000 | 5000 |
| Mote 47 temperature | 3 | 20000 | 5000 |
| Mote 47 humidity | 3 | 20000 | 5000 |
| Housing income | 2 | 18000 | 2000 |
| Housing value | 2 | 18000 | 2000 |
| Housing age | 2 | 18000 | 2000 |

Table 1: Datasets used in our experiments.

| | Exact cost | Tree cost | Speedup | Exact error | Tree error |
|---|---|---|---|---|---|
| Helicopter yaw rate | 14.95 | 0.31 | 47.8 | 0.336 | 0.336 |
| Helicopter x-velocity | 12.37 | 0.41 | 30.3 | 0.594 | 0.595 |
| Helicopter y-velocity | 11.25 | 0.41 | 27.3 | 0.612 | 0.614 |
| Mote 10 temperature | 4.54 | 0.69 | 6.6 | 0.278 | 0.258 |
| Mote 47 temperature | 4.34 | 1.11 | 3.9 | 0.385 | 0.433 |
| Mote 47 humidity | 3.87 | 0.82 | 4.7 | 1.189 | 1.273 |
| Housing income | 2.75 | 0.76 | 3.6 | 0.478 | 0.478 |
| Housing value | 4.47 | 0.51 | 8.8 | 0.496 | 0.496 |
| Housing age | 3.21 | 1.15 | 2.8 | 0.787 | 0.785 |

Table 2: Prediction performance on 9 regression problems. **Exact** uses exact computation of Equation 2. **Tree** is the kd-tree based implementation described in Section 3.2. Cost is the computation time measured in milliseconds per prediction. The error reported is the mean absolute prediction error.

For all experiments, we used the Gaussian RBF kernel

$$K(\boldsymbol{x}, \boldsymbol{x}') = \exp -\frac{\|\boldsymbol{x} - \boldsymbol{x}'\|^2}{2d^2},$$

which is monotonic isotropic, with $d$ and $\sigma$ chosen to be reasonable values for each problem (via cross validation). The $\epsilon$ parameter used in the cutoff rule was set to be $0.001$ for all experiments.

### 5.1 Prediction performance

Our first set of experiments compare the prediction time of the kd-tree algorithm with exact computation, given a precomputed $p$. Our average prediction times are given in Table 2. These numbers include the cost of building the kd-tree (but remain small since the cost is then amortized over all the examples in the test set). As we see, our algorithm runs 2.8-47.8 times faster than exact computation. Further, it incurs only a very small amount of additional error compared to the exact algorithm.

### 5.2 Learning performance

Our second set of experiments examine the running times for learning (i.e., solving the system of Equations 4,) using our kd-tree algorithm for the MVM operation, compared to exact computation. For both approximate and exact MVM, conjugate gradient was used

---

of nearby motes. The housing experiments make use of data collected from the 1990 Census in California. [12] The median income of a block group is predicted from the median house value and average number of rooms per person; the median house value is predicted using median housing age and median income; the median housing age is predicted using median house value and average number of rooms per household.

|  | Exact cost | Tree cost | Speedup | Exact error | Tree error |
|---|---|---|---|---|---|
| Helicopter yaw rate | 22885 | 279 | 82.0 | 0.336 | 0.336 |
| Helicopter x-velocity | 23412 | 619 | 37.9 | 0.594 | 0.595 |
| Helicopter y-velocity | 14341 | 443 | 32.4 | 0.612 | 0.614 |
| Mote 10 temperature | 2071 | 253 | 8.2 | 0.278 | 0.258 |
| Mote 47 temperature | 2531 | 487 | 5.2 | 0.385 | 0.433 |
| Mote 47 humidity | 2121 | 398 | 5.3 | 1.189 | 1.273 |
| Housing income | 1922 | 581 | 3.3 | 0.478 | 0.478 |
| Housing value | 997 | 138 | 7.2 | 0.496 | 0.496 |
| Housing age | 1496 | 338 | 4.4 | 0.787 | 0.785 |

Table 3: Training time on the 9 regression problems. Cost is the computation time measured in seconds.

(with the same number of iterations). Here, we see that our algorithm performs 3.3-82 times faster than exact computation.[3]

## 6  Discussion

### 6.1  Related Work

Multiresolution tree data structures have been used to speed up the computation of a wide variety of machine learning algorithms [9, 5, 7, 14]. GP regression was introduced to the machine learning community by Rasmussen and Williams [19]. The use of CG for efficient first-level inference is described by Gibbs and MacKay [6]. The stability of Krylov subspace iterative solvers (such as CG) with approximate matrix-vector multiplication is discussed in [4].

Sparse approximations to GP inference provide a different way of overcoming the $O(n^3)$ scaling [18, 3, 8], by selecting a representative subset of $D$ of size $d \ll n$. Sparse methods can typically be trained in $O(n\,d^2)$ (including the active forward selection of the subset) and require $O(d)$ prediction time only. In contrast, in our work here we make use of all of the data for prediction, achieving better scaling by exploiting cluster structure in the data through a kd-tree representation.

More closely related to our work is [20], where the MVM primitive is also approximated using a special data structure for $D$. Their approach, called the improved fast Gauss transform (IFGT), partitions the space with a $k$-centers clustering of $D$ and uses a Taylor expansion of the RBF kernel in order to cache repeated computations. The IFGT is limited to the RBF kernel, while our method can be used with all monotonic isotropic kernels. As a topic for future work, we believe it may be possible to apply IFGT's Taylor expansions at each node of the kd-tree's query-dependent multiresolution clustering, to obtain an algorithm that enjoys the best properties of both.

### 6.2  Isotropic Kernels

Recall that an isotropic kernel $K(\boldsymbol{x}, \boldsymbol{x}')$ can be written as a function of the Euclidean distance $r = \|\boldsymbol{x} - \boldsymbol{x}'\|$. While the RBF kernel of the form $\exp(-r^2)$ is the most frequently used isotropic kernel in machine learning, there are many other isotropic kernels to which our method here can be applied without many changes (since the kd-tree cutoff criteria depends on the pairwise Euclidean distances only). An interesting class of kernels is the *Matérn* model (see [17], Sect. 2.10) $K(r) \propto (\alpha r)^\nu K_\nu(\alpha r)$, $\alpha = 2\nu^{1/2}$, where $K_\nu$ is the modified Bessel function of the second kind. The parameter $\nu$ controls the roughness of functions sampled from the process, in that they are $\lfloor \nu \rfloor$ times mean-square differentiable.

For $\nu = 1/2$ we have the "random walk" Ornstein-Uhlenbeck kernel of the form $e^{-\alpha r}$, and the RBF kernel is obtained in the limit $\nu \to \infty$. The RBF kernel forces $u(\cdot)$ to be very smooth, which can lead to bad predictions for training data with partly rough behaviour, and its uncritical usage is therefore discouraged in Geostatistics (where the use of GP models was pioneered). Here, other Matérn kernels are sometimes preferred. We believe that our kd-trees approach holds rich promise for speeding up GPR with other isotropic kernels such the Matérn and Ornstein-Uhlenbeck kernels.

## Footnotes

[1]There are numerous other possible kd-tree splitting criteria. Our criteria is the same as the one used in [9] and [5]

[2]Data for the Helicopter experiments come from an autonomous helicopter flight project, [10] and the three tasks were to model three subdynamics of the helicopter, namely its yaw rate, forward velocity, and lateral velocity one timestep later as a function of the helicopter's current state. The temperature and humidity experiments use data from a sensornet comprising a network of simple sensor motes, [2] and the goal here is to predict the conditions at a mote from the measurements

[3]The errors reported in this table are identical to Table 2, since for the kd-tree results we always trained and made predictions both using the fast approximate method. This gives a more reasonable test of the "end-to-end" use of kd-trees.

## References

[1] Alina Beygelzimer, Sham Kakade, and John Langford. Cover trees for nearest neighbor. (Unpublished manuscript), 2005.

[2] Phil Buonadonna, David Gay, Joseph M. Hellerstein, Wei Hong, and Samuel Madden. Task: Sensor network in a box. In *Proceedings of European Workshop on Sensor Networks*, 2005.

[3] Lehel Csató and Manfred Opper. Sparse on-line Gaussian processes. *Neural Computation*, 14:641–668, 2002.

[4] Nando de Freitas, Yang Wang, Maryam Mahdaviani, and Dustin Lang. Fast krylov methods for n-body learning. In *Advances in NIPS 18*, 2006.

[5] Kan Deng and Andrew Moore. Multiresolution instance-based learning. In *Proceedings of the Twelfth International Joint Conference on Artificial Intellingence*, pages 1233–1239. Morgan Kaufmann, 1995.

[6] Mark N. Gibbs. *Bayesian Gaussian Processes for Regression and Classification*. PhD thesis, University of Cambridge, 1997.

[7] Alexander Gray and Andrew Moore. N-body problems in statistical learning. In *Advances in NIPS 13*, 2001.

[8] N. D. Lawrence, M. Seeger, and R. Herbrich. Fast sparse Gaussian process methods: The informative vector machine. In *Advances in NIPS 15*, pages 609–616, 2003.

[9] Andrew Moore, Jeff Schneider, and Kan Deng. Efficient locally weighted polynomial regression predictions. In *Proceedings of the Fourteenth International Conference on Machine Learning*, pages 236–244. Morgan Kaufmann, 1997.

[10] Andrew Y. Ng, Adam Coates, Mark Diel, Varun Ganapathi, Jamie Schulte, Ben Tse, Eric Berger, and Eric Liang. Inverted autonomous helicopter flight via reinforcement learning. In *International Symposium on Experimental Robotics*, 2004.

[11] Stephen M. Omohundro. Five balltree construction algorithms. Technical Report TR-89-063, International Computer Science Institute, 1989.

[12] R. Kelley Pace and Ronald Barry. Sparse spatial autoregressions. *Statistics and Probability Letters*, 33(3):291–297, May 5 1997.

[13] F.P. Preparata and M. Shamos. *Computational Geometry*. Springer-Verlag, 1985.

[14] Nathan Ratliff and J. Andrew Bagnell. Kernel conjugate gradient. Technical Report CMU-RI-TR-05-30, Robotics Institute, Carnegie Mellon University, June 2005.

[15] Y. Saad. *Iterative Methods for Sparse Linear Systems*. International Thomson Publishing, 1st edition, 1996.

[16] M. Seeger. Gaussian processes for machine learning. *International Journal of Neural Systems*, 14(2):69–106, 2004.

[17] M. Stein. *Interpolation of Spatial Data: Some Theory for Kriging*. Springer, 1999.

[18] Michael Tipping. Sparse Bayesian learning and the relevance vector machine. *Journal of Machine Learning Research*, 1:211–244, 2001.

[19] C. Williams and C. Rasmussen. Gaussian processes for regression. In *Advances in NIPS 8*, 1996.

[20] C. Yang, R. Duraiswami, and L. Davis. Efficient kernel machines using the improved fast Gauss transform. In *Advances in NIPS 17*, pages 1561–1568, 2005.
